# Implicit Mixtures of Restricted Boltzmann Machines

**Vinod Nair and Geoffrey Hinton**
Department of Computer Science, University of Toronto
10 King's College Road, Toronto, M5S 3G5 Canada
{vnair,hinton}@cs.toronto.edu

## Abstract

We present a mixture model whose components are Restricted Boltzmann Machines (RBMs). This possibility has not been considered before because computing the partition function of an RBM is intractable, which appears to make learning a mixture of RBMs intractable as well. Surprisingly, when formulated as a third-order Boltzmann machine, such a mixture model *can* be learned tractably using contrastive divergence. The energy function of the model captures three-way interactions among visible units, hidden units, and a single hidden discrete variable that represents the cluster label. The distinguishing feature of this model is that, unlike other mixture models, the mixing proportions are not explicitly parameterized. Instead, they are defined implicitly via the energy function and depend on all the parameters in the model. We present results for the MNIST and NORB datasets showing that the implicit mixture of RBMs learns clusters that reflect the class structure in the data.

## 1 Introduction

A typical mixture model is composed of a number of separately parameterized density models each of which has two important properties:

1. There is an efficient way to compute the probability density (or mass) of a datapoint under each model.

2. There is an efficient way to change the parameters of each model so as to maximize or increase the sum of the log probabilities it assigns to a set of datapoints.

The mixture is created by assigning a mixing proportion to each of the component models and it is typically fitted by using the EM algorithm that alternates between two steps. The E-step uses property 1 to compute the posterior probability that each datapoint came from each of the component models. The posterior is also called the "responsibility" of each model for a datapoint. The M-step uses property 2 to update the parameters of each model to raise the responsibility-weighted sum of the log probabilities it assigns to the datapoints. The M-step also changes the mixing proportions of the component models to match the proportion of the training data that they are responsible for.

Restricted Boltzmann Machines [5] model binary data-vectors using binary latent variables. They are considerably more powerful than mixture of multivariate Bernoulli models [1] because they allow many of the latent variables to be on simultaneously so the number of alternative latent state vectors is exponential in the number of latent variables rather than being linear in this number as it is with a mixture of Bernoullis. An RBM with $N$ hidden units can be viewed as a mixture of $2^N$ Bernoulli models, one per latent state vector, with a lot of parameter sharing between the $2^N$ component models and with the $2^N$ mixing proportions being implicitly determined by the same parameters.

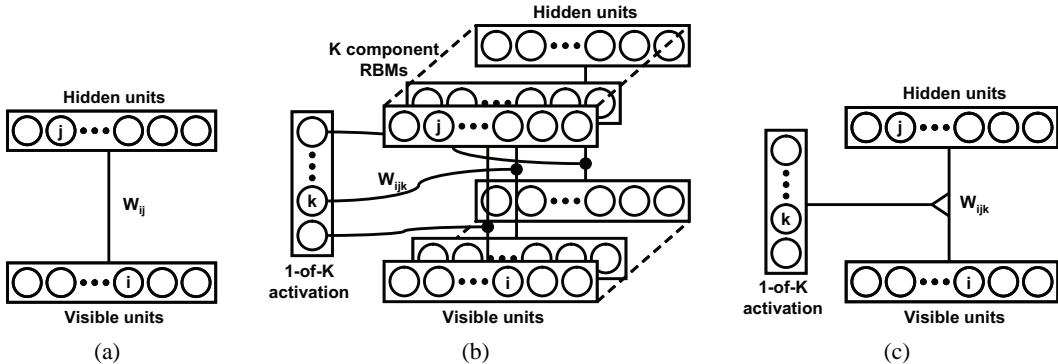

Figure 1: (a) Schematic representation of an RBM, (b) an implicit mixture of RBMs as a third-order Boltzmann machine, (c) schematic representation of an implicit mixture.

It can also be viewed as a product of $N$ "uni-Bernoulli" models (plus one Bernoulli model that is implemented by the visible biases). A uni-Bernoulli model is a mixture of a uniform and a Bernoulli. The weights of a hidden unit define the $i^{th}$ probability in its Bernoulli model as $p_i = \sigma(w_i)$, and the bias, $b$, of a hidden unit defines the mixing proportion of the Bernoulli in its uni-Bernoulli as $\sigma(b)$, where $\sigma(x) = (1 + \exp(-x))^{-1}$.

The modeling power of an RBM can always be increased by increasing the number of hidden units [10] or by adding extra hidden layers [12], but for datasets that contain several distinctly different types of data, such as images of different object classes, it would be more appropriate to use a mixture of RBM's. The mixture could be used to model the raw data or some preprocessed representation that has already extracted features that are shared by different classes. Unfortunately, RBM's cannot easily be used as the components of mixture models because they lack property 1: It is easy to compute the *unnormalized* density that an RBM assigns to a datapoint, but the normalization term is exponentially expensive to compute exactly and even approximating it is extremely time-consuming [11]. There is also no efficient way to modify the parameters of an RBM so that the log probability of the data is guaranteed to increase, but there are good approximate methods [5] so this is not the main problem. This paper describes a way of fitting a mixture of RBM's without explicitly computing the partition function of each RBM.

## 2   The model

We start with the energy function for a Restricted Boltzmann Machine (RBM) and then modify it to define the implicit mixture of RBMs. To simplify the description, we assume that the visible and hidden variables of the RBM are binary. The formulation below can be easily adapted to other types of variables (e.g., see [13]).

The energy function for a Restricted Boltzmann Machine (RBM) is

$$E(\mathbf{v}, \mathbf{h}) = -\sum_{i,j} W_{ij}^R v_i h_j, \tag{1}$$

where $\mathbf{v}$ is a vector of visible (observed) variables, $\mathbf{h}$ is a vector of hidden variables, and $W^R$ is a matrix of parameters that capture pairwise interactions between the visible and hidden variables. Now consider extending this model by including a discrete variable $\mathbf{z}$ with $K$ possible states, represented as a $K$-dimensional binary vector with 1-of-$K$ activation. Defining the energy function in terms of *three-way interactions* among the components of $\mathbf{v}$, $\mathbf{h}$, and $\mathbf{z}$ gives

$$E(\mathbf{v}, \mathbf{h}, \mathbf{z}) = -\sum_{i,j,k} W_{ijk}^I v_i h_j z_k, \tag{2}$$

where $W^I$ is a 3D *tensor* of parameters. Each slice of this tensor along the $\mathbf{z}$-dimension is a matrix that corresponds to the parameters of each of the $K$ component RBMs. The joint distribution for the mixture model is

$$P(\mathbf{v}, \mathbf{h}, \mathbf{z}) = \frac{\exp(-E(\mathbf{v}, \mathbf{h}, \mathbf{z}))}{Z_I}, \tag{3}$$

where

$$Z_I = \sum_{\mathbf{u,g,y}} \exp(-E(\mathbf{u,g,y})) \tag{4}$$

is the partition function of the implicit mixture model. Re-writing the joint distribution in the usual mixture model form gives

$$P(\mathbf{v}) = \sum_{\mathbf{h,z}} P(\mathbf{v,h,z}) = \sum_{k=1}^{K} \sum_{\mathbf{h}} P(\mathbf{v,h}|z_k = 1)P(z_k = 1). \tag{5}$$

Equation 5 defines the implicit mixture of RBMs. $P(\mathbf{v,h}|z_k = 1)$ is the $k^{th}$ component RBM's distribution, with $W^R$ being the $k^{th}$ slice of $W^I$. Unlike in a typical mixture model, the mixing proportion $P(z_k = 1)$ is not a separate parameter in our model. Instead, it is *implicitly* defined via the energy function in equation 2. Changing the bias of the $k^{th}$ unit in $\mathbf{z}$ changes the mixing proportion of the $k^{th}$ RBM, but all of the weights of all the RBM's also influence it. Figure 1 gives a visual description of the implicit mixture model's structure.

## 3   Learning

Given a set of $N$ training cases $\{\mathbf{v}^1, ..., \mathbf{v}^N\}$, we want to learn the parameters of the implicit mixture model by maximizing the log likelihood $L = \sum_{n=1}^{N} \log P(\mathbf{v}^n)$ with respect to $W^I$. We use gradient-based optimization to do this. The expression for the gradient is

$$\frac{\partial L}{\partial W^I} = N \left\langle \frac{\partial E(\mathbf{v,h,z})}{\partial W^I} \right\rangle_{P(\mathbf{v,h,z})} - \sum_{n=1}^{N} \left\langle \frac{\partial E(\mathbf{v}^n,\mathbf{h,z})}{\partial W^I} \right\rangle_{P(\mathbf{h,z}|\mathbf{v}^n)}, \tag{6}$$

where $\langle \rangle_{P()}$ denotes an expectation with respect to the distribution $P()$. The two expectations in equation 6 can be estimated by sample means if unbiased samples can be generated from the corresponding distributions. The conditional distribution $P(\mathbf{h,z}|\mathbf{v}^\alpha)$ is easy to sample from, but sampling the joint distribution $P(\mathbf{v,h,z})$ requires prolonged Gibbs sampling and is intractable in practice. We get around this problem by using the contrastive divergence (CD) learning algorithm [5], which has been found to be effective for training a variety of energy-based models (e.g. [8],[9],[13],[4]).

**Sampling the conditional distributions:** We now describe how to sample the conditional distributions $P(\mathbf{h,z}|\mathbf{v})$ and $P(\mathbf{v}|\mathbf{h,z})$, which are the main operations required for CD learning. The second case is easy: given $z_k = 1$, we select the $k^{th}$ component RBM of the mixture model and then sample from its conditional distribution $P_k(\mathbf{v}|\mathbf{h})$. The bipartite structure of the RBM makes this distribution factorial. So the $i^{th}$ visible unit is drawn independently of the other units from the Bernoulli distribution

$$P(v_i = 1|\mathbf{h}, z_k = 1) = \frac{1}{1 + \exp(-\sum_j W_{ijk}^I h_j)}. \tag{7}$$

Sampling $P(\mathbf{h,z}|\mathbf{v})$ is done in two steps. First, the $K$-way discrete distribution $P(\mathbf{z}|\mathbf{v})$ is computed (see below) and sampled. Then, given $z_k = 1$, we select the $k^{th}$ component RBM and sample from its conditional distribution $P_k(\mathbf{h}|\mathbf{v})$. Again, this distribution is factorial, and the $j^{th}$ hidden unit is drawn from the Bernoulli distribution

$$P(h_j = 1|\mathbf{v}, z_k = 1) = \frac{1}{1 + \exp(-\sum_i W_{ijk}^I v_i)}. \tag{8}$$

To compute $P(\mathbf{z}|\mathbf{v})$ we first note that

$$P(z_k = 1|\mathbf{v}) \propto \exp(-F(\mathbf{v}, z_k = 1)), \tag{9}$$

where the *free energy* $F(\mathbf{v}, z_k = 1)$ is given by

$$F(\mathbf{v}, z_k = 1) = -\sum_j \log(1 + \exp(\sum_i W_{ijk}^I v_i)). \tag{10}$$

If the number of possible states of $\mathbf{z}$ is small enough, then it is practical to compute the quantity $F(\mathbf{v}, z_k = 1)$ for every $k$ by brute-force. So we can compute

$$P(z_k = 1|\mathbf{v}) = \frac{\exp(-F(\mathbf{v}, z_k = 1))}{\sum_l \exp(-F(\mathbf{v}, z_l = 1))}. \tag{11}$$

Equation 11 defines the *responsibility* of the $k^{th}$ component RBM for the data vector $\mathbf{v}$.

**Contrastive divergence learning:** Below is a summary of the steps in the CD learning for the implicit mixture model.

1. For a training vector $\mathbf{v}_+$, pick a component RBM by sampling the responsibilities $P(z_k = 1|\mathbf{v}_+)$. Let $l$ be the index of the selected RBM.

2. Sample $\mathbf{h}_+ \sim P_l(\mathbf{h}|\mathbf{v}_+)$.

3. Compute the outer product $\mathbf{D}_l^+ = \mathbf{v}_+ \mathbf{h}_+^T$.

4. Sample $\mathbf{v}_- \sim P_l(\mathbf{v}|\mathbf{h}_+)$.

5. Pick a component RBM by sampling the responsibilities $P(z_k = 1|\mathbf{v}_-)$. Let $m$ be the index of the selected RBM.

6. Sample $\mathbf{h}_- \sim P_m(\mathbf{h}|\mathbf{v}_-)$.

7. Compute the outer product $\mathbf{D}_m^- = \mathbf{v}_- \mathbf{h}_-^T$.

Repeating the above steps for a mini-batch of $N_b$ training cases results in two sets of outer products for each component $k$ in the mixture model: $S_k^+ = \{\mathbf{D}_{k1}^+, ..., \mathbf{D}_{kM}^+\}$ and $S_k^- \{\mathbf{D}_{k1}^-, ..., \mathbf{D}_{kL}^-\}$. Then the approximate likelihood gradient (averaged over the mini-batch) for the $k^{th}$ component RBM is

$$\frac{1}{N_b} \frac{\partial L}{\partial W_k^I} \approx \frac{1}{N_b} \left( \sum_{i=1}^{M} \mathbf{D}_{ki}^+ - \sum_{j=1}^{L} \mathbf{D}_{kj}^- \right). \tag{12}$$

Note that to compute the outer products $\mathbf{D}^+$ and $\mathbf{D}^-$ for a given training vector, the component RBMs are selected through *two separate stochastic picks*. Therefore the sets $S_k^+$ and $S_k^-$ need not be of the same size because the choice of the mixture component can be different for $\mathbf{v}_+$ and $\mathbf{v}_-$.

**Scaling free energies with a temperature parameter:** In practice, the above learning algorithm causes all the training cases to be captured by a single component RBM, and the other components to be left unused. This is because free energy is an unnormalized quantity that can have very different numerical scales across the RBMs. One RBM may happen to produce much smaller free energies than the rest because of random differences in the initial parameter values, and thus end up with high responsibilities for most training cases. Even if all the component RBMs are initialized to the exact same initial parameter values, the problem can still arise after a few noisy weight updates. The solution is to use a temperature parameter $T$ when computing the responsibilities:

$$P(z_k = 1|\mathbf{v}) = \frac{\exp(-F(\mathbf{v}, z_k = 1)/T)}{\sum_l \exp(-F(\mathbf{v}, z_l = 1)/T)}. \tag{13}$$

By choosing a large enough $T$, we can make sure that random scale differences in the free energies do not lead to the above collapse problem. One possibility is to start with a large $T$ and then gradually anneal it as learning progresses. In our experiments we found that using a constant $T$ works just as well as annealing, so we keep it fixed.

## 4   Results

We apply the implicit mixture of RBMs to two datasets, MNIST [1] and NORB [7]. MNIST is a set of handwritten digit images belonging to ten different classes (the digits 0 to 9). NORB contains stereo-pair images of 3D toy objects taken under different lighting conditions and viewpoints. There are five classes of objects in this set (*human*, *car*, *plane*, *truck* and *animal*). We use MNIST mainly as a sanity check, and most of our results are for the much more difficult NORB dataset.

**Evaluation method:** Since computing the exact partition function of an RBM is intractable, it is not possible to directly evaluate the quality of our mixture model's fit to the data, e.g., by computing

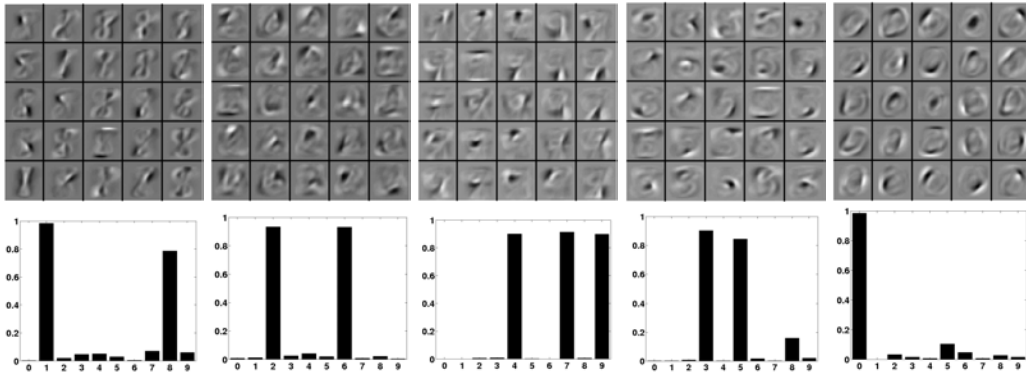

Figure 2: Features of the mixture model with five component RBMs trained on all ten classes of MNIST images.

the log probability of a test set under the model. Recently it was shown that Annealed Importance Sampling can be used to tractably approximate the partition function of an RBM [11]. While this is an attractive option to consider in future work, for this paper we use the computationally cheaper approach of evaluating the model by using it in a classification task. Classification accuracy is then used as an indirect quantitative measure of how good the model is.

A reasonable evaluation criterion for a mixture modelling algorithm is that it should be able to find clusters that are mostly 'pure' with respect to class labels. That is, the set of data vectors that a particular mixture component has high responsibilities for should have the same class label. So it should be possible to accurately predict the class label of a given data vector from the responsibilities of the different mixture components for that vector. Once a mixture model is fully trained, we evaluate it by training a classifier that takes as input the responsibilities of the mixture components for a data vector and predicts its class label. The goodness of the mixture model is measured by the test set prediction accuracy of this classifier.

## 4.1   Results for MNIST

Before attempting to learn a good mixture model of the whole MNIST dataset, we tried two simpler modeling tasks. First, we fitted an implicit mixture of two RBM's with 100 hidden units each to an unlabelled dataset consisting of 4,000 twos and 4,000 threes. As we hoped, almost all of the two's were modelled by one RBM and almost all of the threes by the other. On 2042 held-out test cases, there were only 24 errors when an image was assigned the label of the most probable RBM. This compares very favorably with logistic regression which needs 8000 labels in addition to the images and gives 36 errors on the test set even when using a penalty on the squared weights whose magnitude is set using a validation set. Logistic regression also gives a good indication of the performance that could be expected from fitting a mixture of two Gaussians with a shared covariance matrix, because logistic regression is equivalent to fitting such a mixture discriminatively.

We then tried fitting an implicit mixture model with only five component RBMs, each with 25 hidden units, to the entire training set. We purposely make the model very small so that it is possible to visually inspect the features and the responsibilities of the component RBMs and understand what each component is modelling. This is meant to qualitatively confirm that the algorithm can learn a sensible clustering of the MNIST data. (Of course, the model will have poor classification accuracy as there are more classes than clusters, so it will merge multiple classes into a single cluster.) The features of the component RBMs are shown in figure 2 (top row). The plots in the bottom row show the fraction of training images for each of the ten classes that are hard-assigned to each component. The learning algorithm has produced a sensible mixture model in that visually similar digit classes are combined under the same mixture component. For example, ones and eights require many similar features, so they are captured with a single RBM (leftmost in fig. 2). Similarly, images of fours, sevens, and nines are all visually similar, and they are modelled together by one RBM (middle of fig. 2).

We have also trained larger models with many more mixture components. As the number of components increase, we expect the model to partition the image space more finely, with the different components specializing on various sub-classes of digits. If they specialize in a way that respects the class boundaries, then their responsibilities for a data vector will become a better predictor of its class label.

The component RBMs use binary units both in the visible and hidden layers. The image dimensionality is 784 ($28 \times 28$ pixels). We have tried various settings for the number of mixture components (from 20 to 120 in steps of 20) and a component's hidden layer size (50, 100, 200, 500). Classification accuracy increases with more components, until 80 components. Additional components give slightly worse results. The hidden layer size is set to 100, but 200 and 500 also produce similar accuracies. Out of the 60,000 training images in MNIST, we use 50,000 to train the mixture model and the classifier, and the remaining 10,000 as a validation set for early stopping. The final models are then tested on a separate test set of 10,000 images.

Once the mixture model is trained, we train a logistic regression classifier to predict the class label from the responsibilities[2]. It has as many inputs as there are mixture components, and a ten-way softmax over the class labels at the output. With 80 components, there are only $80 \cdot 10 + 10 = 810$ parameters in the classifier (including the 10 output biases). In our experiments, classification accuracy is consistently and significantly higher when *unnormalized* responsibilities are used as the classifier input, instead of the actual posterior probabilities of the mixture components given a data vector. These unnormalized values have no proper probabilistic interpretation, but nevertheless they allow for better classification, so we use them in all our experiments.

Table 1: MNIST Test set error rates.

| Logistic regression classifier input | % Test error |
|---|---|
| Unnormalized responsibilities | 3.36% |
| Pixels | 7.28% |

Table 1 shows the classification error rate of the resulting classifier on the MNIST test set. As a simple baseline comparison, we train a logistic regression classifier that predicts the class label from the raw pixels. This classifier has $784 \cdot 10 + 10 = 7850$ parameters and yet the mixture-based classifier has less than half the error rate. The unnormalized responsibilities therefore contain a significant amount of information about the class labels of the images, which indicates that the implicit mixture model has learned clusters that mostly agree with the class boundaries, even though it is not given any class information during training.

## 4.2 Results for NORB

NORB is a much more difficult dataset than MNIST because the images are of very different classes of 3D objects (instead of 2D patterns) shown from different viewpoints and under various lighting conditions. The pixels are also no longer binary-valued, but instead span the grayscale range $[0, 255]$. So binary units are no longer appropriate for the visible layer of the component RBMs. Gaussian visible units have previously been shown to be effective for modelling grayscale images [6], and therefore we use them here. See [6] for details about Gaussian units. As in that paper, the variance of the units is fixed to 1, and only their means are learned.

Learning an RBM with Gaussian visible units can be slow, as it may require a much greater number of weight updates than an equivalent RBM with binary visible units. This problem becomes even worse in our case since a large number of RBMs have to be trained simultaneously. We avoid it by first training a single RBM with Gaussian visible units and binary hidden units on the raw pixel data, and then treating the activities of its hidden layer as pre-processed data to which the implicit mixture model is applied. Since the hidden layer activities of the pre-processing RBM are binary, the mixture model can now be trained efficiently with binary units in the visible layer[3]. Once trained, the low-level RBM acts as a fixed pre-processing step that converts the raw grayscale images into

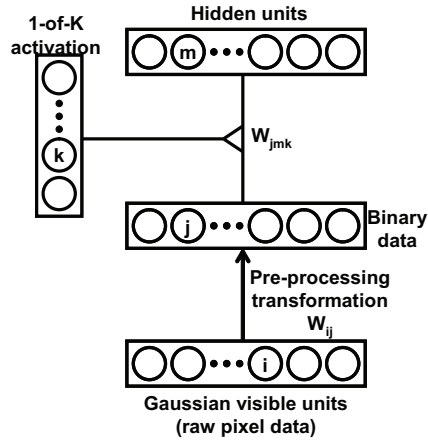

Figure 3: Implicit mixture model used for MNORB.

binary vectors. Its parameters are not modified further when training the mixture model. Figure 3 shows the components of the complete model.

A difficulty with training the implicit mixture model (or any other mixture model) on NORB is that the 'natural' clusters in the dataset correspond to the six lighting conditions instead of the five object classes. The objects themselves are small (in terms of area) relative to the background, while lighting affects the entire image. Any clustering signal provided by the object classes will be weak compared to the effect of large lighting changes. So we simplify the dataset slightly by normalizing the lighting variations across images. Each image is multiplied by a scalar such that all images have the same average pixel value. This significantly reduces the interference of the lighting on the mixture learning[4]. Finally, to speed up experiments, we subsample the images from $96 \times 96$ to $32 \times 32$ and use only one image of the stereo pair. We refer to this dataset as 'Modified NORB' or 'MNORB'. It contains 24,300 training images and an equal number of test images. From the training set, 4,300 are set aside as a validation set for early stopping.

We use 2000 binary hidden units for the preprocessing RBM, so the input dimensionality of the implicit mixture model is 2000. We have tried many different settings for the number of mixture components and the hidden layer size of the components. The best classification results are given by 100 components, each with 500 hidden units. This model has about $100 \cdot 500 \cdot 2000 = 10^8$ parameters, and takes about 10 days to train on an Intel Xeon 3Ghz processor.

Table 2 shows the test set error rates for a logistic regression classifier trained on various input representations. Mixture of Factor Analyzers (MFA) [3] is similar to the implicit mixture of RBMs in that it also learns a clustering while simultaneously learning a latent representation per cluster component. But it is a directed model based on linear-Gaussian representations, and it can be learned tractably by maximizing likelihood with EM. We train MFA on the raw pixel data of MNORB. The MFA model that gives the best classification accuracy (shown in table 2) has 100 component Factor Analyzers with 100 factors each. (Note that simply making the number of learnable parameters equal is not enough to match the capacities of the different models because RBMs use binary latent representations, while FAs use continuous representations. So we cannot strictly control for capacity when comparing these models.)

A mixture of multivariate Bernoulli distributions (see *e.g.* section 9.3.3 of [2]) is similar to an implicit mixture model whose component RBMs have no hidden units and only visible biases as trainable parameters. The differences are that a Bernoulli mixture is a directed model, it has explicitly parameterized mixing proportions, and maximum likelihood learning with EM is tractable. We train this model with 100 components on the activation probabilities of the preprocessing RBM's hidden units. The classification error rate for this model is shown in table 2.

Table 2: MNORB Test set error rates for a logistic regression classifier with different types of input representations.

| Logistic regression classifier input | % Test error |
|---|---|
| Unnormalized responsibilities computed by the implicit mixture of RBMs | 14.65% |
| Probabilities computed by the transformation $W_{ij}$ in fig 3 (i.e. the *pre-processed representation*) | 16.07% |
| Raw pixels | 20.60% |
| Unnormalized responsibilities of an MFA model trained on the pre-processed representation in fig 3 | 22.65% |
| Unnormalized responsibilities of an MFA model trained on raw pixels | 24.57% |
| Unnormalized responsibilities of a Mixture of Bernoullis model trained on the pre-processed representation in fig 3 | 28.53% |

These results show that the implicit mixture of RBMs has learned clusters that reflect the class structure in the data. By the classification accuracy criterion, the implicit mixture is also better than MFA. The results also confirm that the lack of explicitly parameterized mixing proportions does not prevent the implicit mixture model from discovering interesting cluster structure in the data.

## 5   Conclusions

We have presented a tractable formulation of a mixture of RBMs. That such a formulation is even possible is a surprising discovery. The key insight here is that the mixture model can be cast as a third-order Boltzmann machine, provided we are willing to abandon explicitly parameterized mixing proportions. Then it can be learned tractably using contrastive divergence. As future work, it would be interesting to explore whether these ideas can be extended to modelling time-series data.

## Footnotes

[1] A multivariate Bernoulli model consists of a set of probabilities, one per component of the binary data vector.

[2]Note that the mixture model parameters are kept fixed when training the classifier, so the learning of the mixture model is entirely unsupervised.

[3]We actually use the real-valued probabilities of the hidden units as the data, and we also use real-valued probabilities for the reconstructions. On other tasks, the learning gives similar results using binary values sampled from these real-valued probabilities but is slower.

[4]The normalization does not completely remove lighting information from the data. A logistic regression classifier can still predict the lighting label with 18% test set error when trained and tested on normalized images, compared to 8% error for unnormalized images.

## References

[1] Mnist database, http://yann.lecun.com/exdb/mnist/.

[2] C. M. Bishop. *Pattern Recognition and Machine Learning*. Springer, 2006.

[3] Z. Ghahramani and G. E. Hinton. The em algorithm for mixtures of factor analyzers. *Technical Report CRG-TR-96-1, Dept. of Computer Science, University of Toronto*, 1996.

[4] X. He, R. S. Zemel, and M. A. Carreira-Perpinan. Multiscale conditional random fields for image labeling. In *CVPR*, pages 695–702, 2004.

[5] G. E. Hinton. Training products of experts by minimizing contrastive divergence. *Neural Computation*, 14(8):1711–1800, 2002.

[6] G. E. Hinton and R. Salakhutdinov. Reducing the dimensionality of data with neural networks. *Science*, 313:504–507, 2006.

[7] Y. LeCun, F. J. Huang, and L. Bottou. Learning methods for generic object recognition with invariance to pose and lighting. In *CVPR*, Washington, D.C., 2004.

[8] S. Roth and M. J. Black. Fields of experts: A framework for learning image priors. In *CVPR*, pages 860–867, 2005.

[9] S. Roth and M. J. Black. Steerable random fields. In *ICCV*, 2007.

[10] N. Le Roux and Y. Bengio. Representational power of restricted boltzmann machines and deep belief networks. *Neural Computation*, To appear.

[11] R. Salakhutdinov and I. Murray. On the quantitative analysis of deep belief networks. In *ICML*, Helsinki, 2008.

[12] I. Sutskever and G. E. Hinton. Deep narrow sigmoid belief networks are universal approximators. *Neural Computation*, To appear.

[13] M. Welling, M. Rosen-Zvi, and G. E. Hinton. Exponential family harmoniums with an application to information retrieval. In *NIPS 17*, 2005.

